# Bayesian nonparametric models for bipartite graphs

**François Caron**
INRIA
IMB - University of Bordeaux
Talence, France
Francois.Caron@inria.fr

## Abstract

We develop a novel Bayesian nonparametric model for random bipartite graphs. The model is based on the theory of completely random measures and is able to handle a potentially infinite number of nodes. We show that the model has appealing properties and in particular it may exhibit a power-law behavior. We derive a posterior characterization, a generative process for network growth, and a simple Gibbs sampler for posterior simulation. Our model is shown to be well fitted to several real-world social networks.

## 1 Introduction

The last few years have seen a tremendous interest in the study, understanding and statistical modeling of complex networks [14, 6]. A network is a set if items, called vertices, with connections between them, called edges. In this article, we shall focus on bipartite networks, also known as two-mode, affiliation or collaboration networks [16, 17]. In bipartite networks, items are divided into two different types $A$ and $B$, and only connections between items of different types are allowed. Examples of this kind can be found in movie actors co-starring the same movie, scientists co-authoring a scientific paper, internet users posting a message on the same forum, people reading the same book or listening to the same song, members of the boards of company directors sitting on the same board, etc. Following the readers-books example, we will refer to items of type $A$ as *readers* and items of type $B$ as *books*. An example of bipartite graph is shown on Figure 1(b). An important summarizing quantity of a bipartite graph is the degree distribution of readers (resp. books) [14]. The degree of a vertex in a network is the number of edges connected to that vertex. Degree distributions of real-world networks are often strongly non-Poissonian and exhibit a power-law behavior [15].

A bipartite graph can be represented by a set of binary variables $(z_{ij})$ where $z_{ij} = 1$ if reader $i$ has read book $j$, 0 otherwise. In many situations, the number of available books may be very large and potentially unknown. In this case, a Bayesian nonparametric (BNP) approach can be sensible, by assuming that the pool of books is infinite. To formalize this framework, it will then be convenient to represent the bipartite graph by a collection of atomic measures $Z_i$, $i = 1, \ldots, n$ with

$$Z_i = \sum_{j=1}^{\infty} z_{ij}\delta_{\theta_j} \tag{1}$$

where $\{\theta_j\}$ is the set of books and typically $Z_i$ only has a finite set of non-zero $z_{ij}$ corresponding to books reader $i$ has read. Griffiths and Ghahramani [8, 9] have proposed a BNP model for such binary random measures. The so-called Indian Buffet Process (IBP) is a simple generative process for the conditional distribution of $Z_i$ given $Z_1, \ldots, Z_{i-1}$. Such process can be constructed by considering that the binary measures $Z_i$ are i.i.d. from some random measure drawn from a beta process [19, 10]. It has found several applications for inferring hidden causes [20], choices [7] or features [5]. Teh and Görür [18] proposed a three-parameter extension of the IBP, named stable IBP, that enables to

model a power-law behavior for the degree distribution of books. Although more flexible, the stable IBP still induces a Poissonian distribution for the degree of readers.

In this paper, we propose a novel Bayesian nonparametric model for bipartite graphs that addresses some of the limitations of the stable IBP, while retaining computational tractability. We assume that each book $j$ is assigned a positive popularity parameter $w_j > 0$. This parameter measures the popularity of the book, larger weights indicating larger probability to be read. Similarly, each reader $i$ is assigned a positive parameter $\gamma_i$ which represents its ability to read books. The higher $\gamma_i$, the more books the reader $i$ is willing to read. Given the weights $w_j$ and $\gamma_i$, reader $i$ reads book $j$ with probability $1 - \exp(-\gamma_i w_j)$. We will consider that the weights $w_j$ and/or $\gamma_i$ are the points of a Poisson process with a given Lévy measure. We show that depending on the choice of the Lévy measure, a power-law behavior can be obtained for the degree distribution of books and/or readers. Moreover, using a set of suitably chosen latent variables, we can derive a generative process for network growth, and an efficient Gibbs sampler for approximate inference. We provide illustrations of the fit of the proposed model on several real-world bipartite social networks. Finally, we discuss some potentially useful extensions of our work, in particular to latent factor models.

## 2 Statistical Model

### 2.1 Completely Random Measures

We first provide a brief overview of completely random measures (CRM) [12, 13] before describing the BNP model for bipartite graphs in Section 2.2. Let $\Theta$ be a measurable space. A CRM is a random measure $G$ such that for any collection of disjoint measurable subsets $A_1, \ldots, A_n$ of $\Theta$, the random masses of the subsets $G(A_1), \ldots, G(A_n)$ are independent. CRM can be decomposed into a sum of three independent parts: a non-random measure, a countable collection of atoms with fixed locations, and a countable collection of atoms with randoms masses at random locations. In this paper, we will be concerned with models defined by CRMs with random masses at random locations, i.e. $G = \sum_{j=1}^{\infty} w_j \delta_{\theta_j}$. The law of $G$ can be characterized in terms of a Poisson process over the point set $\{(w_j, \theta_j), j = 1, \ldots, \infty\} \subset \mathbb{R}^+ \times \Theta$. The mean measure $\Lambda$ of this Poisson process is known as the Lévy measure. We will assume in the following that the Lévy measure decomposes as a product of two non-atomic densities, i.e. that $G$ is a homogeneous CRM $\Lambda(dw, d\theta) = \lambda(w)h(\theta)dwd\theta$ with $h : \Theta \to [0, +\infty)$ and $\int_{\Theta} h(\theta)d\theta = 1$. It implies that the locations of the atoms in $G$ are independent of the masses, and are i.i.d. from $h$, while the masses are distributed according to a Poisson process over $\mathbb{R}^+$ with mean intensity $\lambda$. We will further assume that the total mass $G(\Theta) = \sum_{j=1}^{\infty} w_j$ is positive and finite with probability one, which is guaranteed if the following conditions are satisfied

$$\int_0^{\infty} \lambda(w)dw = \infty \quad \text{and} \quad \int_0^{\infty} (1 - \exp(-w))\lambda(w)dw < \infty \tag{2}$$

and note $g(x)$ its probability density function evaluated at $x$. We will refer to $\lambda$ as the Lévy intensity in the following, and to $h$ as the base density of $G$, and write $G \sim \text{CRM}(\lambda, h)$. We will also note

$$\psi_\lambda(t) = -\log \mathbb{E}\left[\exp(-tG(\Theta))\right] = \int_0^{\infty} (1 - \exp(-tw))\lambda(w)dw \tag{3}$$

$$\widetilde{\psi}_\lambda(t, b) = \int_0^{\infty} (1 - \exp(-tw))\lambda(w)\exp(-bw)dw \tag{4}$$

$$\kappa(n, z) = \int_0^{\infty} \lambda(w)w^n e^{-zw}dw \tag{5}$$

As a notable particular example of CRM, we can mention the generalized gamma process (GGP) [1], whose Lévy intensity is given by
$$\lambda(w) = \frac{\alpha}{\Gamma(1 - \sigma)}w^{-\sigma-1}e^{-w\tau}$$

GGP encompasses the gamma process ($\sigma = 0$), the inverse Gaussian process ($\sigma = 0.5$) and the stable process ($\tau = 0$) as special cases. Table **??** in supplementary material provides the expressions of $\lambda$, $\psi$ and $\kappa$ for these processes.

## 2.2 A Bayesian nonparametric model for bipartite graphs

Let $G \sim \text{CRM}(\lambda, h)$ where $\lambda$ satisfies conditions (2). A draw $G$ takes the form

$$G = \sum_{j=1}^{\infty} w_j \delta_{\theta_j} \tag{6}$$

where $\{\theta_j\}$ is the set of books and $\{w_j\}$ the set of popularity parameters of books. For $i = 1, \dots, n$, let consider the latent exponential process

$$V_i = \sum_{j=1}^{\infty} v_{ij} \delta_{\theta_j} \tag{7}$$

defined for $j = 1, \dots, \infty$ by $v_{ij}|w_j \sim \text{Exp}(w_j \gamma_i)$ where $\text{Exp}(a)$ denotes the exponential distribution of rate $a$. The higher $w_j$ and/or $\gamma_i$, the lower $v_{ij}$. We then define the binary process $Z_i$ conditionally on $V_i$ by

$$Z_i = \sum_{j=1}^{\infty} z_{ij} \delta_{\theta_j} \quad \text{with} \quad \begin{cases} z_{ij} = 1 & \text{if } v_{ij} < 1 \\ z_{ij} = 0 & \text{otherwise} \end{cases} \tag{8}$$

By integrating out the latent variables $v_{ij}$ we clearly have $p(z_{ij} = 1|w_j, \gamma_i) = 1 - \exp(-\gamma_i w_j)$.

**Proposition 1** $Z_i$ is marginally characterized by a Poisson process over the point set $\{(\theta_j^*), j = 1, \dots, \infty\} \subset \Theta$, of intensity measure $\psi_\lambda(\gamma_i) h(\theta^*)$. Hence, the total mass $Z_i(\Theta) = \sum_{j=1}^{\infty} z_{ij}$, which corresponds to the total number of books read by reader $i$ is finite with probability one and admits a $\text{Poisson}(\psi_\lambda(\gamma_i))$ distribution, where $\psi_\lambda(z)$ is defined in Equation (3), while the locations $\theta_j^*$ are i.i.d. from $h$.

The proof, which makes use of Campbell's theorem for point processes [13] is given in supplementary material. As an example, for the gamma process we have $Z_i(\Theta) \sim \text{Poisson}\left(\alpha \log\left(1 + \frac{\gamma_i}{\tau}\right)\right)$.

It will be useful in the following to introduce a censored version of the latent process $V_i$, defined by

$$U_i = \sum_{j=1}^{\infty} u_{ij} \delta_{\theta_j} \tag{9}$$

where $u_{ij} = \min(v_{ij}, 1)$, for $i = 1, \dots, n$ and $j = 1, \dots, \infty$. Note that $Z_i$ can be obtained deterministically from $U_i$.

## 2.3 Characterization of the conditional distributions

The conditional distribution of $G$ given $Z_1, \dots, Z_n$ cannot be obtained in closed form[1]. We will make use of the latent process $U_i$. In this section, we derive the formula for the conditional laws $P(U_1, \dots, U_n|G)$, $P(U_1, \dots, U_n)$ and $P(G|U_1, \dots, U_n)$. Based on these results, we derive in Section 2.4 a generative process and in Section 2.5 a Gibbs sampler for our model, that both rely on the introduction of these latent variables.

Assume that $K$ books $\{\theta_1, \dots, \theta_K\}$ have appeared. We write $K_i = Z_i(\Theta) = \sum_{j=1}^{\infty} z_{ij}$ the degree of reader $i$ (number of books read by reader $i$) and $m_j = \sum_{i=1}^{n} Z_i(\{\theta_j\}) = \sum_{i=1}^{n} z_{ij}$ the degree of book $j$ (number of people having read book $j$). The conditional likelihood of $U_1, \dots U_n$ given $G$ is given by

$$P(U_1, \dots U_n|G) = \prod_{i=1}^{n} \left\{ \left[ \prod_{j=1}^{K} \gamma_i^{z_{ij}} w_j^{z_{ij}} \exp\left(-\gamma_i w_j u_{ij}\right) \right] \exp\left(-\gamma_i G(\Theta \backslash \{\theta_1, \dots, \theta_K\})\right) \right\}$$

$$= \left( \prod_{i=1}^{n} \gamma_i^{K_i} \right) \left[ \prod_{j=1}^{K} w_j^{m_j} \exp\left(-w_j \sum_{i=1}^{n} \gamma_i(u_{ij} - 1)\right) \right] \exp\left(-\left(\sum_{i=1}^{n} \gamma_i\right) G(\Theta)\right) \tag{10}$$

**Proposition 2** *The marginal distribution $P(U_1, \ldots U_n)$ is given by*

$$P(U_1, \ldots U_n) = \left( \prod_{i=1}^{n} \gamma_i^{K_i} \right) \exp \left[ -\psi_\lambda \left( \sum_{i=1}^{n} \gamma_i \right) \right] \prod_{j=1}^{K} h(\theta_j) \kappa \left( m_j, \sum_{i=1}^{n} \gamma_i u_{ij} \right) \qquad (11)$$

*where $\psi_\lambda$ and $\kappa$ are resp. defined by Eq. (3) and (5).*

**Proof.** The proof, detailed in supplementary material, is obtained by an application of the Palm formula for CRMs [3, 11], and is the same as that of Theorem 1 in [2]. ∎

**Proposition 3** *The conditional distribution of $G$ given the latent processes $U_1, \ldots U_n$ can be expressed as*

$$G = G^* + \sum_{j=1}^{K} w_j \delta_{\theta_j} \qquad (12)$$

*where $G^*$ and $(w_j)$ are mutually independent with*

$$G^* \sim \mathrm{CRM}(\lambda^*, h) \qquad\qquad \lambda^*(w) = \lambda(w) \exp \left( -w \sum_{i=1}^{n} \gamma_i \right) \qquad (13)$$

*and the masses are*

$$P(w_j | rest) = \frac{\lambda(w_j) w_j^{m_j} \exp \left( -w_j \sum_{i=1}^{n} \gamma_i U_{ij} \right)}{\kappa(m_j, \sum_{i=1}^{n} \gamma_i u_{ij})} \qquad (14)$$

**Proof.** The proof, based on the application of the Palm formula and detailed in supplementary material, is the same as that of Theorem 2 in [2]. ∎

In the case of the GGP, $G^*$ is still a GGP of parameters $(\alpha^* = \alpha, \sigma^* = \sigma, \tau^* = \tau + \sum_{i=1}^{n} \gamma_i)$, while the $w_j$'s are conditionally gamma distributed, i.e.

$$w_j | \text{rest} \sim \mathrm{Gamma} \left( m_j - \sigma, \tau + \sum_{i=1}^{n} \gamma_i u_{ij} \right)$$

**Corollary 4** *The predictive distribution of $Z_{n+1}$ given the latent processes $U_1, \ldots, U_n$ is given by*

$$Z_{n+1} = Z_{n+1}^* + \sum_{j=1}^{K} z_{n+1,j} \delta_{\theta_j}$$

*where the $z_{n+1,j}$ are independent of $Z_{n+1}^*$ with*

$$z_{n+1,j} | U \sim \mathrm{Ber} \left( 1 - \frac{\kappa(m_j, \tau + \gamma_{n+1} + \sum_{i=1}^{n} \gamma_i u_{ij})}{\kappa(m_j, \tau + \sum_{i=1}^{n} \gamma_i u_{ij})} \right)$$

*where* Ber *is the Bernoulli distribution and $Z_{n+1}^*$ is a homogeneous Poisson process over $\Theta$ of intensity measure $\psi_{\lambda^*}(\gamma_{n+1}) h(\theta)$.*

For the GGP, we have

$$Z_{n+1}^*(\Theta) \sim \begin{cases} \mathrm{Poisson} \left( \frac{\alpha}{\sigma} \left[ \left( \tau + \sum_{i=1}^{n+1} \gamma_i \right)^\sigma - \left( \tau + \sum_{i=1}^{n} \gamma_i \right)^\sigma \right] \right) & \text{if } \sigma \neq 0 \\ \mathrm{Poisson} \left( \alpha \log \left( 1 + \frac{\gamma_{n+1}}{\tau + \sum_{i=1}^{n} \gamma_i} \right) \right) & \text{if } \sigma = 0 \end{cases}$$

and $\quad z_{n+1,j} | U \sim \mathrm{Ber} \left( 1 - \left( 1 + \frac{\gamma_{n+1}}{\tau + \sum_{i=1}^{n} \gamma_i u_{ij}} \right)^{-m_j + \sigma} \right).$

Finally, we consider the distribution of $u_{n+1,j} | z_{n+1,j} = 1, u_{1:n,j}$. This is given by

$$p(u_{n+1,j} | z_{n+1,j} = 1, u_{1:n,j}) \propto \kappa(m_j + 1, u_{n+1,j} \gamma_{n+1} + \sum_{i=1}^{n} \gamma_i u_{ij}) \mathbf{1}_{u_{n+1,j} \in [0,1]} \qquad (15)$$

In supplementary material, we show how to sample from this distribution by the inverse cdf method for the GGP.

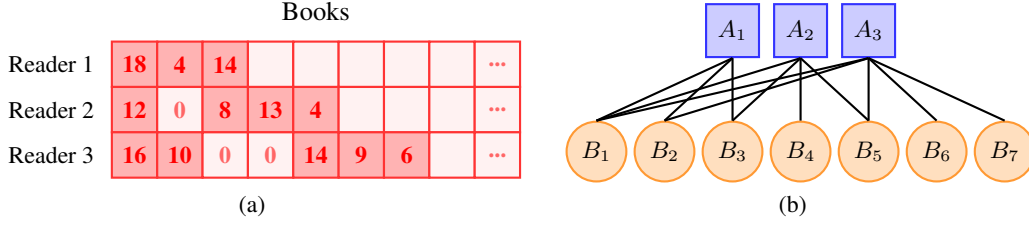

Figure 1: Illustration of the generative process described in Section 2.4.

## 2.4 A generative process

In this section we describe the generative process for $Z_i$ given $(U_1, \ldots, U_{i-1})$, $G$ being integrated out. This reinforcement process, where popular books will be more likely to be picked, is reminiscent of the generative process for the beta-Bernoulli process, popularized under the name of the Indian buffet process [8]. Let $x_{ij} = -\log(u_{ij}) \geq 0$ be latent positive scores.

Consider a set of $n$ readers who successively enter into a library with an infinite number of books. Each reader $i = 1, \ldots n$, has some interest in reading quantified by a positive parameter $\gamma_i > 0$. The first reader picks a number $K_1 \sim \text{Poisson}(\psi_\lambda(\gamma_1))$ books. Then he assigns a positive score $x_{1j} = -\log(u_{1j}) > 0$ to each of these books, where $u_{1j}$ is drawn from distribution (15).

Now consider that reader $i$ enters into the library, and knows about the books read by previous readers and their scores. Let $K$ be the total number of books chosen by the previous $i - 1$ readers, and $m_j$ the number of times each of the $K$ books has been read. Then for each book $j = 1, \ldots, K$, reader $i$ will choose this book with probability

$$1 - \frac{\kappa(m_j, \tau + \gamma_i + \sum_{k=1}^{i-1} \gamma_k u_{kj})}{\kappa(m_j, \tau + \sum_{k=1}^{i-1} \gamma_k u_{kj})}$$

and then will choose an additional number of $K_i^+$ books where

$$K_i^+ \sim \text{Poisson}\left(\widetilde{\psi}_\lambda\left(\gamma_i, \sum_{k=1}^{i-1} \gamma_k\right)\right)$$

Reader $i$ will then assign a score $x_{ij} = -\log u_{ij} > 0$ to each book $j$ he has read, where $u_{ij}$ is drawn from (15). Otherwise he will set the default score $x_{ij} = 0$. This generative process is illustrated in Figure 1 together with the underlying bipartite graph . In Figure 2 are represented draws from this generative process with a GGP with parameters $\gamma_i = 2$ for all $i$, $\tau = 1$, and different values for $\alpha$ and $\sigma$.

## 2.5 Gibbs sampling

From the results derived in Proposition 3, a Gibbs sampler can be easily derived to approximate the posterior distribution $P(G, U|Z)$. The sampler successively updates $U$ given $(w, G^*(\Theta))$ then $(w, G^*(\Theta))$ given $U$. We present here the conditional distributions in the GGP case. For $i = 1, \ldots, n, j = 1, \ldots, K$, set $u_{ij} = 1$ if $z_{ij} = 0$, otherwise sample

$$u_{ij}|z_{ij}, w_j, \gamma_i \sim \text{rExp}(\gamma_i w_j, 1)$$

where $\text{rExp}(\lambda, a)$ is the right-truncated exponential distribution of pdf $\lambda \exp(-\lambda x)/(1 - \exp(-\lambda a))\mathbf{1}_{x \in [0,a]}$ from which we can sample exactly. For $j = 1, \ldots, K$, sample

$$w_j|U, \gamma_i \sim \text{Gamma}\left(m_j - \sigma, \tau + \sum_{i=1}^{n} \gamma_i u_{ij}\right)$$

and the total mass $G^*(\Theta)$ follows a distribution $g^*(w) \propto g(w) \exp\left(-w \sum_{i=1}^{n} \gamma_i\right)$ where $g(w)$ is the distribution of $G(\Theta)$. In the case of the GGP, $g^*(w)$ is an exponentially tilted stable distribution for which exact samplers exist [4]. In the particular case of the gamma process, we have the simple update $G^*(\Theta) \sim \text{Gamma}\left(\alpha, \tau + \sum_{i=1}^{n} \gamma_i\right)$.

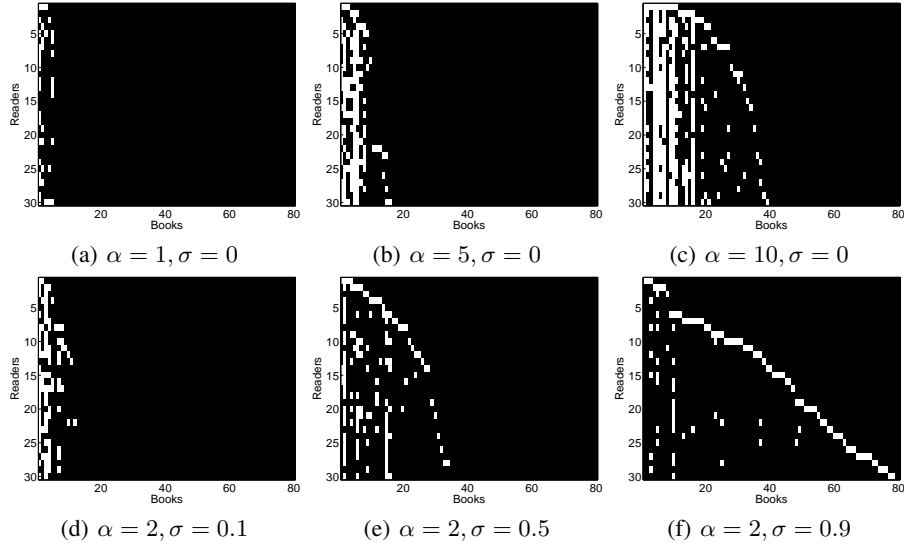

Figure 2: Realisations from the generative process of Section 2.4 with a GGP of parameters $\gamma = 2$, $\tau = 1$ and various values of $\alpha$ and $\sigma$.

## 3 Update of $\gamma_i$ and other hyperparameters

We may also consider the weight parameters $\gamma_i$ to be unknown and estimate them from the graph. We can assign a gamma prior $\gamma_i \sim \text{Gamma}(a_\gamma, b_\gamma)$ with parameters $(a_\gamma > 0, b_\gamma > 0)$ and update it conditionally on other variables with

$$\gamma_i | G, U \sim \text{Gamma} \left( a_\gamma + \sum_{j=1}^{K} z_{ij}, b_\gamma + \sum_{j=1}^{K} w_j u_{ij} + G^*(\Theta) \right)$$

In this case, the marginal distribution of $Z_i(\Theta)$, hence the degree distribution of books, follows a continuous mixture of Poisson distributions, which offers more flexibility in the modelling.

We may also go a step further and consider that there is an infinite number of readers with weights $\gamma_i$ associated to a given CRM $\Gamma \sim \text{CRM}(\lambda_\gamma, h_\gamma)$ and a measurable space of readers $\widetilde{\Theta}$. We then have $\Gamma = \sum_{i=1}^{\infty} \gamma_i \delta_{\widetilde{\theta}_i}$. This provides a lot of flexibility in the modelling of the distribution of the degree of readers, allowing in particular to obtain a power-law behavior, as shown in Section 5. We focus here on the case where $\Gamma$ is drawn from a generalized gamma process of parameters $(\alpha_\gamma, \sigma_\gamma, \tau_\gamma)$ for simplicity. Conditionally on $(w, G^*(\Theta), U)$, we have $\Gamma = \Gamma^* + \sum_{i=1}^{n} \gamma_i \delta_{\widetilde{\theta}_i}$ where for $i = 1, \dots, n$,

$$\gamma_i | G, U \sim \text{Gamma} \left( \sum_{j=1}^{K} z_{ij} - \sigma_\gamma, \tau + \sum_{j=1}^{K} w_j u_{ij} + G^*(\Theta) \right)$$

and $\Gamma^* \sim \text{CRM}(\lambda_\gamma^*, h_\gamma)$ with $\lambda_\gamma^*(\gamma) = \lambda_\gamma(\gamma) \exp\left(-\gamma \left( \sum_{j=1}^{K} w_j + G^*(\Theta) \right)\right)$. In this case, the update for $(w, G^*)$ conditional on $(U, \gamma, \Gamma(\widetilde{\Theta}))$ is now for $j = 1, \dots, K$

$$w_j | U, \Gamma \sim \text{Gamma} \left( m_j - \sigma, \tau + \sum_{i=1}^{n} \gamma_i u_{ij} + \Gamma^*(\widetilde{\Theta}) \right)$$

and $G^* \sim \text{CRM}(\lambda^*, h)$ with $\lambda^*(w) = \lambda(w) \exp\left(-w \left( \sum_{i=1}^{n} \gamma_i + \Gamma^*(\widetilde{\Theta}) \right)\right)$. Note that there is now symmetry in the treatment of books/readers. For the scale parameter $\alpha$ of the GGP, we can assign a gamma prior $\alpha \sim \text{Gamma}(a_\alpha, b_\alpha)$ and update it with $\alpha | \gamma \sim \text{Gamma}\left(a_\alpha + K, b_\alpha + \psi_\lambda \left( \sum_{i=1}^{n} \gamma_i + \Gamma^*(\widetilde{\Theta}) \right)\right)$. Other parameters of the GGP can be updated using a Metropolis-Hastings step.

## 4 Discussion

**Power-law behavior.** We now discuss some of the properties of the model, in the case of the GGP. The total number of books read by $n$ readers is $O(n^\sigma)$. Moreover, for $\sigma > 0$, the degree distribution follows a power-law distribution: asymptotically, the proportion of books read by $m$ readers is $O(m^{-1-\sigma})$ (details in supplementary material). These results are similar to those of the stable IBP [18]. However, in our case, a similar behavior can be obtained for the degree distribution of readers when assigning a GGP to it, while it will always be Poisson for the stable IBP.

**Connection to IBP.** The stable beta process [18] is a particular case of our construction, obtained by setting weights $\gamma_i = \gamma$ and Lévy measure

$$\lambda(w) = \alpha \frac{\Gamma(1+c)}{\Gamma(1-\sigma)\Gamma(c+\sigma)} \gamma (1 - e^{-\gamma w})^{-\sigma-1} e^{-\gamma w(c+\sigma)} \quad (16)$$

The proof is obtained by a change of variable from the Lévy measure of the stable beta process.

**Extensions to latent factor models.** So far, we have assumed that the binary matrix $Z$ was observed. The proposed model can also be used as a prior for latent factor models, similarly to the IBP. As an example of the potential usefulness of our model compared to IBP, consider the extraction of features from time series of different lengths. Longer time series are more likely to exhibit more features than shorter ones, and it is sensible in this case to assume different weights $\gamma_i$. In a more general setting, we may want $\gamma_i$ to depend on a set of metadata associated to reader $i$. Inference for latent factor models is described in supplementary material.

## 5 Illustrations on real-world social networks

We now consider estimating the parameters of our model and evaluating its predictive performance on six bipartite social networks of various sizes. We first provide a short description of these networks. The dataset 'Boards' contains information about members of the boards of Norwegian companies sitting at the same board in August 2011[2]. 'Forum' is a forum network about web users contributing to the same forums[3]. 'Books' concerns data collected from the Book-Crossing community about users providing ratings on books[4] where we extracted the bipartite network from the ratings. 'Citations' is the co-authorship network based on preprints posted to Condensed Matter section of ArXiv between 1995 and 1999 [15]. 'Movielens100k' contains information about users rating particular movies[5] from which we extracted the bipartite network. Finally, 'IMDB' contains information about actors co-starring a movie[6]. The sizes of the different networks are given in Table 1.

| Dataset | $n$ | $K$ | Edges | S-IBP | SG | IG | GGP |
|---|---|---|---|---|---|---|---|
| Board | 355 | 5766 | 1746 | **9.82** | 8.3 | -145.1 | -68.6 |
|  |  |  |  | **(29.8)** | (30.8) | (81.9) | (31.9) |
| Forum | 899 | 552 | 7089 | -6.7e3 | -6.7e3 | **-5.5e3** | -5.6e3 |
| Books | 5064 | 36275 | 49997 | 83.1 | 214 | **4.6e4** | 4.4e4 |
| Citations | 16726 | 22016 | 58595 | -3.7e4 | -3.7e4 | **-3.1e4** | -3.4e4 |
| Movielens100k | 943 | 1682 | 100000 | -6.7e4 | -6.7e4 | **-5.5e4** | **-5.5e4** |
| IMDB | 28088 | 178074 | 341313 | -1.5e5 | -1.5e5 | **-1.1e5** | **-1.1e5** |

Table 1: Size of the different datasets and test log-likelihood of four different models.

We evaluate the fit of four different models on these datasets. First, the stable IBP [18] with parameters $(\alpha_{IBP}, \tau_{IBP}, \sigma_{IBP})$ (S-IBP). Second, our model where the parameter $\gamma$ is the same over different readers, and is assigned a flat prior (SG). Third our model where each $\gamma_i \sim \mathrm{Gamma}(a_\gamma, b_\gamma)$ where $(a_\gamma, b_\gamma)$ are unknown parameters with flat improper prior (IG). Finally, our model with a GGP model for $\gamma_i$, with parameters $(\alpha_\gamma, \sigma_\gamma, \tau_\gamma)$ (GGP). We divide each dataset between a training

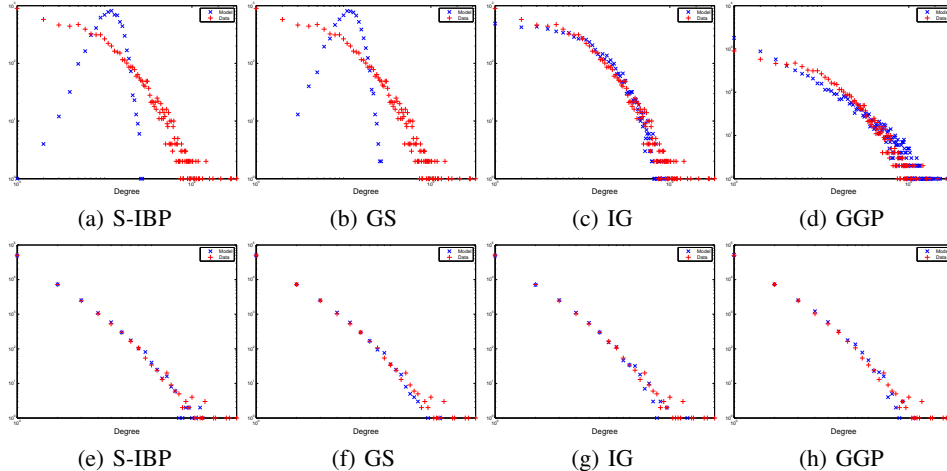

Figure 3: Degree distributions for movies (a-d) and actors (e-h) for the IMDB movie-actor dataset with four different models. Data are represented by red plus and samples from the model by blue crosses.

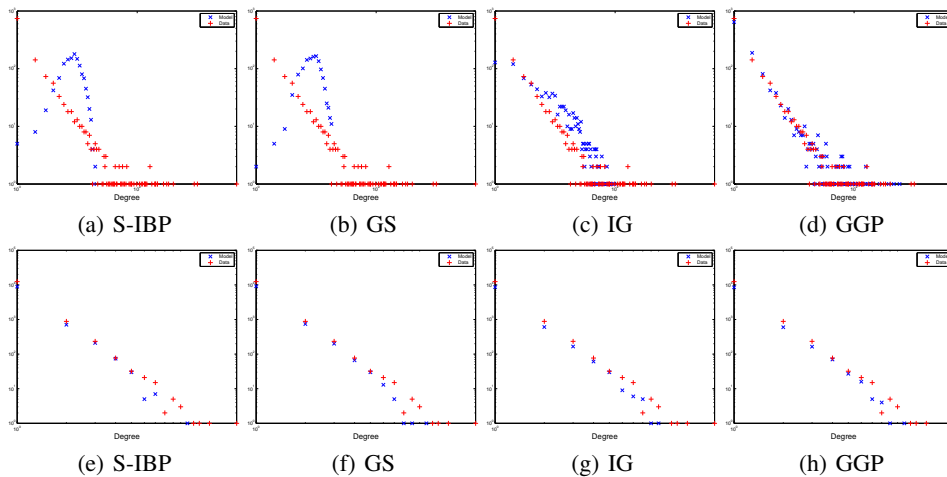

Figure 4: Degree distributions for readers (a-d) and books (e-h) for the BX books dataset with four different models. Data are represented by red plus and samples from the model by blue crosses.

set containing $3/4$ of the readers and a test set with the remaining. For each model, we approximate the posterior mean of the unknown parameters (respectively $(\alpha_{IBP}, \tau_{IBP}, \sigma_{IBP})$, $\gamma$, $(a_\gamma, b_\gamma)$ and $(\alpha_\gamma, \sigma_\gamma, \tau_\gamma)$ for S-IBP, SG, IG and GGP) given the training network with a Gibbs sampler with 10000 burn-in iterations then 10000 samples; then we evaluate the log-likelihood of the estimated model on the test data. For GGP, we use $\alpha_\gamma^{test} = \widehat{\alpha}_\gamma/3$ to take into account the different sample sizes. For 'Boards', we do 10 replications with random permutations given the small sample size and report standard deviation together with mean value. Table 1 shows the results over the different networks for the different models. Typically, S-IBP and SG give very similar results. This is not surprising, as they share the same properties, i.e. Poissonian degree distribution for readers and power-law degree distribution for books. Both methods perform better solely on the Board dataset, where the Poisson assumption on the number of people sitting on the same board makes sense. On all the other datasets, IG and GGP perform better and similarly, with slightly better performances for IG. These two models are better able to capture the power-law distribution of the degrees of readers. These properties are shown on Figures 3 and 4 which resp. give the empirical degree distributions of the test network and a draw from the estimated models, for the IMDB dataset and the Books dataset. It is clearly seen that the four models are able to capture the power-law behavior of the degree distribution of actors (Figure 3(e-h)) or books (Figure 4(e-h)). However, only IG and GGP are able to capture the power-law behavior of the degree distribution of movies (Figure 3(a-d)) or readers (Figure 4(a-d)).

## Footnotes

[1] In the case where $\gamma_i = \gamma$, it is possible to derive $P(Z_1, \dots, Z_n)$ and $P(Z_{n+1}|Z_1, \dots, Z_n)$ where the random measure $G$ and the latent variables $U$ are marginalized out. This particular case is described in supplementary material.

[2]Data can be downloaded from http://www.boardsandgender.com/data.php

[3]Data for the forum and citation datasets can be downloaded from http://toreopsahl.com/datasets/

[4]http://www.informatik.uni-freiburg.de/ cziegler/BX/

[5]The dataset can be downloaded from http://www.grouplens.org

[6]The dataset can be downloaded from http://www.cise.ufl.edu/research/sparse/matrices/Pajek/IMDB.html

# References

[1] A. Brix. Generalized gamma measures and shot-noise Cox processes. *Advances in Applied Probability*, 31(4):929–953, 1999.

[2] F. Caron and Y. W. Teh. Bayesian nonparametric models for ranked data. In *Neural Information Processing Systems (NIPS)*, 2012.

[3] D.J. Daley and D. Vere-Jones. *An introduction to the theory of point processes*. Springer Verlag, 2008.

[4] L. Devroye. Random variate generation for exponentially and polynomially tilted stable distributions. *ACM Transactions on Modeling and Computer Simulation (TOMACS)*, 19(4):18, 2009.

[5] E.B. Fox, E.B. Sudderth, M.I. Jordan, and A.S. Willsky. Sharing features among dynamical systems with beta processes. In *Advances in Neural Information Processing Systems*, volume 22, pages 549–557, 2009.

[6] A. Goldenberg, A.X. Zheng, S.E. Fienberg, and E.M. Airoldi. A survey of statistical network models. *Foundations and Trends in Machine Learning*, 2(2):129–233, 2010.

[7] D. Görür, F. Jäkel, and C.E. Rasmussen. A choice model with infinitely many latent features. In *Proceedings of the 23rd international conference on Machine learning*, pages 361–368. ACM, 2006.

[8] T Griffiths and Z. Ghahramani. Infinite latent feature models and the Indian buffet process. In *NIPS*, 2005.

[9] T. Griffiths and Z. Ghahramani. The Indian buffet process: an introduction and review. *Journal of Machine Learning Research*, 12(April):1185–1224, 2011.

[10] N.L. Hjort. Nonparametric bayes estimators based on beta processes in models for life history data. *The Annals of Statistics*, 18(3):1259–1294, 1990.

[11] L.F. James, A. Lijoi, and I. Prünster. Posterior analysis for normalized random measures with independent increments. *Scandinavian Journal of Statistics*, 36(1):76–97, 2009.

[12] J.F.C. Kingman. Completely random measures. *Pacific Journal of Mathematics*, 21(1):59–78, 1967.

[13] J.F.C. Kingman. *Poisson processes*, volume 3. Oxford University Press, USA, 1993.

[14] M.E.J. Newman. The structure and function of complex networks. *SIAM review*, pages 167–256, 2003.

[15] M.E.J. Newman, S.H. Strogatz, and D.J. Watts. Random graphs with arbitrary degree distributions and their applications. *Physical Review E*, 64(2):26118, 2001.

[16] M.E.J. Newman, D.J. Watts, and S.H. Strogatz. Random graph models of social networks. *Proceedings of the National Academy of Sciences*, 99:2566, 2002.

[17] J.J. Ramasco, S.N. Dorogovtsev, and R. Pastor-Satorras. Self-organization of collaboration networks. *Physical review E*, 70(3):036106, 2004.

[18] Y.W. Teh and D. Görür. Indian buffet processes with power-law behavior. In *NIPS*, 2009.

[19] R. Thibaux and M. Jordan. Hierarchical beta processes and the Indian buffet process. In *International Conference on Artificial Intelligence and Statistics*, volume 11, pages 564–571, 2007.

[20] F. Wood, T.L. Griffiths, and Z. Ghahramani. A non-parametric Bayesian method for inferring hidden causes. In *Proceedings of the Conference on Uncertainty in Artificial Intelligence*, volume 22, 2006.

